# Exploiting spatial overlap to efficiently compute appearance distances between image windows

**Bogdan Alexe**
ETH Zurich

**Viviana Petrescu**
ETH Zurich

**Vittorio Ferrari**
ETH Zurich

## Abstract

We present a computationally efficient technique to compute the distance of high-dimensional appearance descriptor vectors between image windows. The method exploits the relation between appearance distance and spatial overlap. We derive an upper bound on appearance distance given the spatial overlap of two windows in an image, and use it to bound the distances of many pairs between two images. We propose algorithms that build on these basic operations to efficiently solve tasks relevant to many computer vision applications, such as finding all pairs of windows between two images with distance smaller than a threshold, or finding the single pair with the smallest distance. In experiments on the PASCAL VOC 07 dataset, our algorithms accurately solve these problems while greatly reducing the number of appearance distances computed, and achieve larger speedups than approximate nearest neighbour algorithms based on trees [18] and on hashing [21]. For example, our algorithm finds the most similar pair of windows between two images while computing only 1% of all distances on average.

## 1 Introduction

Computing the appearance distance between two windows is a fundamental operation in a wide variety of computer vision techniques. Algorithms for weakly supervised learning of object classes [7, 11, 16] typically compare large sets of windows between images trying to find recurring patterns of appearance. Sliding-window object detectors based on kernel SVMs [13, 24] compute appearance distances between the support vectors and a large number of windows in the test image. In human pose estimation, [22] computes the color histogram dissimilarity between many candidate windows for lower and upper arms. In image retrieval the user can search a large image database for a query object specified by an image window [20]. Finally, many tracking algorithms [4, 5] compare a window around the target object in the current frame to all windows in a surrounding region of the next frame.

In most cases one is not interested in computing the distance between *all* pairs of windows from two sets, but in a small subset of low distances, such as all pairs below a given threshold, or the single best pair. Because of this, computer vision researchers often rely on efficient nearest neighbour algorithms [2, 6, 10, 17, 18, 21]. Exact nearest neighbour algorithms organize the appearance descriptors into trees which can be efficiently searched [17]. However, these methods work well only for descriptors of small dimensionality $n$ (typically $n < 20$), and their speedup vanishes for larger $n$ (e.g. the popular GIST descriptor [19] has $n = 960$). Locality sensitive hashing (LSH [2, 10, 21]) techniques hash the descriptors into bins, so that similar descritors are mapped to the same bins with high probability. LSH is typically used for efficiently finding *approximate* nearest neighbours in high dimensions [2, 6].

All the above methods consider windows only as points in *appearance space*. However, windows exist also as points in the *geometric space* defined as their 4D coordinates in the image they lie in. In this geometric space, a natural distance between two windows is their *spatial overlap* (fig. 1). In this paper we propose to take advantage of an important relation between the geometric and appearance spaces: the apparance distance between two windows decreases as their spatial overlap increases. We derive an upper bound on the appearance distance between two windows in the same image,

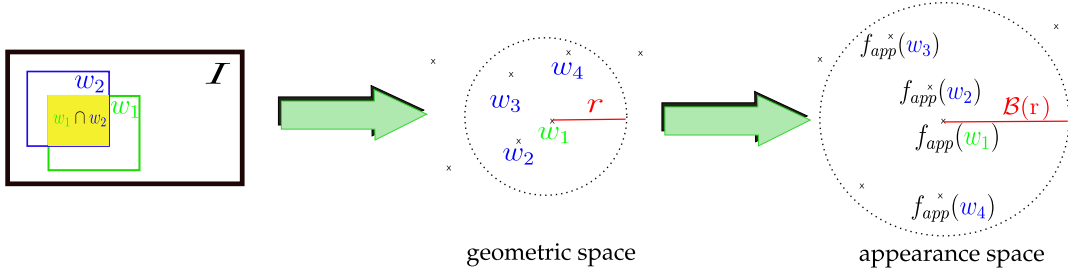

<div align="center">geometric space        appearance space</div>

Fig. 1: **Relation between spatial overlap and appearance distance.** *Windows $w_1$, $w_2$ in an image $I$ are embedded in geometric space and in appearance space. All windows overlapping more than $r$ with $w_1$ are at most at distance $\mathcal{B}(r)$ in appearance space. The bound $\mathcal{B}(r)$ decreases as overlap increases (i.e. $r$ decreases).*

given their spatial overlap (sec. 2). We then use this bound in conjuction with the triangle inequality to bound the appearance distances of many pairs of windows between two images, given the distance of just one pair. Building on these basic operations, we design algorithms to efficiently find all pairs with distance smaller than a threshold (sec. 3) and to find the single pair with the smallest distance (sec. 4).

The techniques we propose reduce computation by minimizing the *number of times* appearance distances are computed. They are complementary to methods for reducing the *cost of computing one distance*, such as dimensionality reduction [15] or Hamming embeddings [14, 23].

We experimentally demonstrate in sec. 5 that the proposed algorithms accurately solve the above problems while greatly reducing the number of appearance distances computed. We compare to approximate nearest neighbour algorithms based on trees [18], as well as on the recent LSH technique [21]. The results show our techniques outperform them in the setting we consider, where the datapoints are embedded in a space with additional overlap structure.

## 2    Relation between spatial overlap and appearance distance

Windows $w$ in an image $I$ are emdebbed in two spaces at the same time (fig. 1). In geometric space, $w$ is represented by its 4 spatial coordinates (e.g. $x, y$ center, width, height). The distance between two windows is defined based on their *spatial overlap* $o(w_1, w_2) = \frac{|w_1 \cap w_2|}{|w_1 \cup w_2|} \in [0, 1]$, where $\cap$ denotes the area of the intersection and $\cup$ the area of the union. In appearance space, $w$ is represented by a high dimensional vector describing the pixel pattern inside it, as computed by a function $f_{app}(w) : I \to R^n$ (e.g. the GIST descriptor has $n = 960$ dimensions). In appearance space, two windows are compared using a distance $d(f_{app}(w_1), f_{app}(w_2))$.

Two overlapping windows $w_1, w_2$ in an image $I$ share the pixels contained in their intersection (fig. 1). The spatial overlap of the two windows correlates with the proportion of common pixels input to $f_{app}$ when computing the descriptor for each window. In general, $f_{app}$ varies smoothly with the geometry of $w$, so that windows of similar geometry are close in appearance space. Consequently, the spatial overlap $o$ and appearance distance $d$ are *related*. In this paper we exploit this relation to derive an upper bound $\mathcal{B}(o(w_1, w_2))$ on the appearance distance between two overlapping windows.

We present here the general form of the bound $\mathcal{B}$, its main properties and explain why it is useful. In subsections 2.1 and 2.2 we derive the actual bound itself. To simplify the notation we use $d(w_1, w_2)$ to denote the appearance distance $d(f_{app}(w_1), f_{app}(w_2))$. We refer to it simply as *distance* and we say *overlap* for spatial overlap. The upper bound $\mathcal{B}$ is a function of the overlap $o(w_1, w_2)$, and has the following property

$$d(w_1, w_2) \le \mathcal{B}(o(w_1, w_2)) \quad \forall w_1, w_2 \tag{1}$$

Moreover, $\mathcal{B}$ is a monotonic decreasing function

$$\mathcal{B}(o_1) \le \mathcal{B}(o_2) \quad \forall o_1 \ge o_2 \tag{2}$$

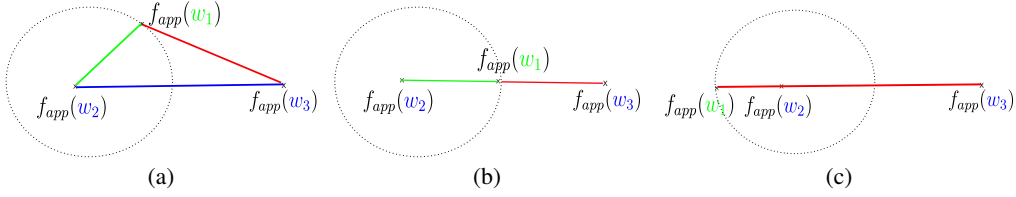

(a)&emsp;&emsp;&emsp;&emsp;&emsp;&emsp;&emsp;&emsp;(b)&emsp;&emsp;&emsp;&emsp;&emsp;&emsp;&emsp;&emsp;(c)

Fig. 2: **Triangle inequality in appearance space.** *The triangle inequality* (4) *holds for any three points* $\mathrm{f_{app}}(w_1)$, $\mathrm{f_{app}}(w_2)$ *and* $\mathrm{f_{app}}(w_3)$ *in appearance space. (a) General case; (b) Lower bound case:* $|\mathrm{d}(w_1, w_2) - \mathrm{d}(w_2, w_3)| = \mathrm{d}(w_1, w_3)$; *(c) Upper bound case:* $\mathrm{d}(w_1, w_3) = \mathrm{d}(w_1, w_2) + \mathrm{d}(w_2, w_3)$.

This property means $\mathcal{B}$ continuously decreases as overlap increases. Therefore, all pairs of windows within an overlap radius $r$ (i.e. $\mathrm{o}(w_1, w_2) \geq r$) have distance below $\mathcal{B}(r)$ (fig. 1)

$$\mathrm{d}(w_1, w_2) \leq \mathcal{B}(\mathrm{o}(w_1, w_2)) \leq \mathcal{B}(r) \quad \forall w_1, w_2, \ \mathrm{o}(w_1, w_2) \geq r \tag{3}$$

As defined above, $\mathcal{B}$ bounds the appearance distance between two windows in the same image. Now we show how it can be used to derive a bound on the distances between windows in two different images $I^1, I^2$. Given two windows $w_1, w_2$ in $I^1$ and a window $w_3$ in $I^2$, we use the triangle inequality to derive (fig. 2)

$$|\mathrm{d}(w_1, w_2) - \mathrm{d}(w_2, w_3)| \ \leq \ \mathrm{d}(w_1, w_3) \ \leq \ \mathrm{d}(w_1, w_2) + \mathrm{d}(w_2, w_3) \tag{4}$$

Using the bound $\mathcal{B}$ in eq. (4) we obtain

$$\max(0, \ \mathrm{d}(w_2, w_3) - \mathcal{B}(\mathrm{o}(w_1, w_2))) \ \leq \ \mathrm{d}(w_1, w_3) \ \leq \ \mathcal{B}(\mathrm{o}(w_1, w_2)) + \mathrm{d}(w_2, w_3) \tag{5}$$

Eq. (5) delivers lower and upper bounds for $\mathrm{d}(w_1, w_3)$ *without explicitly computing it* (given that $\mathrm{d}(w_2, w_3)$ and $\mathrm{o}(w_1, w_2)$ are known). These bounds will form the basis of our algorithms for reducing the number of times the appearance distance is computed when solving two classic tasks (sec. 3 and 4).

In the next subsection we estimate $\mathcal{B}$ for arbitrary window descriptors (e.g. color histograms, bag of visual words, GIST [19], HOG [8]) from a set of images (no human annotation required). In subsection 2.2 we derive exact bounds in closed form for histogram descriptors (e.g. color histograms, bag of visual words [25]).

### 2.1 Statistical bounds for arbitrary window descriptors

We estimate $\mathcal{B}_\alpha$ from training data so that eq. (1) holds with probability $\alpha$

$$P(\ \mathrm{d}(w_1, w_2) \leq \mathcal{B}_\alpha(\mathrm{o}(w_1, w_2))\ ) = \alpha \quad \forall w_1, w_2 \tag{6}$$

$\mathcal{B}_\alpha$ is estimated from a set of $M$ training images $\mathcal{I} = \{I^m\}$. For each image $I^m$ we sample $N$ windows $\{w_i^m\}$, and then compute for all window pairs their overlap $o_{ij}^m = \mathrm{o}(w_i^m, w_j^m)$ and distance $d_{ij}^m = \mathrm{d}(w_i^m, w_j^m)$. The overall training dataset $\mathcal{D}$ is composed of $(o_{ij}^m, d_{ij}^m)$ for every window pair

$$\mathcal{D} = \{\ (o_{ij}^m, d_{ij}^m) \mid k \in \{1, M\}, \ i, j \in \{1, N\}\} \tag{7}$$

We now quantize the overlap values into 100 bins and estimate $\mathcal{B}_\alpha(o)$ for each bin $o$ separately. For a bin $o$, we consider the set $\mathcal{D}_o$ of all distances $d_{ij}^m$ for which $o_{ij}^m$ is in the bin. We choose $\mathcal{B}_\alpha(o)$ as the $\alpha$-quantile of $\mathcal{D}(o)$ (fig. 3a)

$$\mathcal{B}_\alpha(o) = q_\alpha(\mathcal{D}_o) \tag{8}$$

$\mathcal{B}_1(o)$ is the largest distance $d_{ij}^m$ for which $o_{ij}^m$ is in bin $o$. Fig. 3a shows the binned distance-overlap pairs and the bound $\mathcal{B}_{0.95}$ for GIST descriptors [19]. The data comes from 100 windows sampled from more than 1000 images (details in sec. 5). Each column of this matrix is roughly Gaussian distributed, and its mean continuously decreases with increasing overlap, confirming our assumptions about the relation between overlap and distance (sec. 2). In particular, note how the mean distance decrease fastest for 50% to 80% overlap.

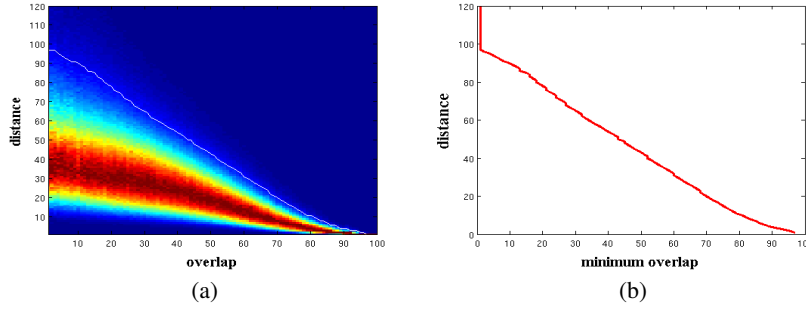

Fig. 3: **Estimating $\mathcal{B}_{0.95}(o)$ and $o_{min}(\epsilon)$.** *(a) The estimated $\mathcal{B}_{0.95}(o)$ (white line) for the GIST [19] appearance descriptor. (b) Using $\mathcal{B}_{0.95}(o)$ we derive $o_{min}(\epsilon)$.*

Given a window $w_1$ and a distance $\epsilon$ we can use $\mathcal{B}_\alpha$ to find windows $w_2$ overlapping with $w_1$ that are at most distance $\epsilon$ from $w_1$. This will be used extensively by our algorithms presented in secs. 3 and 4. From $\mathcal{B}_\alpha$ we can derive what is the smallest overlap $o_{min}(\epsilon)$ so that all pairs of windows overlapping more than $o_{min}(\epsilon)$ have distance smaller than $\epsilon$ (with probability more than $\alpha$). Formally

$$P(\,\mathrm{d}(w_1, w_2) \leq \epsilon\,) \;\geq\; \alpha \quad \forall w_1, w_2, o(w_1, w_2) \geq o_{min}(\epsilon) \tag{9}$$

and $o_{min}(\epsilon)$ is defined as the smallest overlap $o$ for which the bound is smaller than $\epsilon$ (fig. 3b)

$$o_{min}(\epsilon) = \min\{o \mid \mathcal{B}_\alpha(o) \leq \epsilon\} \tag{10}$$

## 2.2 Exact bounds for histogram descriptors

The statistical bounds of the previous subsection can be estimated from images for any appearance descriptor. In contrast, in this subsection we derive *exact* bounds in closed form for histogram descriptors (e.g. color histograms, bag of visual words [25]). Our derivation applies to $L_1$-normalized histograms and the $\chi^2$ distance. For simplicity of presentation, we assume every pixel contributes one feature to the histogram of the window (as in color histograms). The derivation is very similar for features computed on another regular grid (e.g. dense SURF bag-of-words [11]). We present here the main idea behind the bound and give the full derivation in the supplementary material [1].

The upper bound $\mathcal{B}$ for two windows $w_1$ and $w_2$ corresponds to the limit case where the three regions $w_1 \cap w_2$, $w_1 \setminus w_2$ and $w_2 \setminus w_1$ contain three disjoint sets of colors (or visual word in general). Therefore, the upper bound $\mathcal{B}$ is

$$\mathcal{B}(w_1, w_2) = \frac{|w_1 \setminus w_2|}{|w_1|} + \frac{|w_2 \setminus w_1|}{|w_2|} + |w_1 \cap w_2| \cdot \frac{\left(\frac{1}{|w_1|} - \frac{1}{|w_2|}\right)^2}{\frac{1}{|w_1|} + \frac{1}{|w_2|}} \tag{11}$$

Expressing the terms in (11) based on the windows overlap $o = o(w_1, w_2) = \frac{|w_1 \cap w_2|}{|w_1 \cup w_2|}$, we obtain a closed form for the upper bound $\mathcal{B}$ that depends only on $o$

$$\mathcal{B}(w_1, w_2) = \mathcal{B}(o(w_1, w_2)) = \mathcal{B}(o) = 2 - 4 \cdot \frac{o}{o+1} \tag{12}$$

In practice, this exact bound is typically much looser than its corresponding statistical bound learned from data (sec. 2.1). Therefore, we use the statistical bound for the experiments in sec. 5.

## 3 Efficiently computing all window pairs with distance smaller than $\epsilon$

In this section we present an algorithm to efficiently find all pairs of windows with distance smaller than a threshold $\epsilon$ between two images $I^1, I^2$. Formally, given an input set of windows $\mathcal{W}^1 = \{w_i^1\}$ in image $I^1$ and a set $\mathcal{W}^2 = \{w_j^2\}$ in image $I^2$, the algorithm should return the set of pairs $\mathcal{P}_\epsilon = \{\,(w_i^1, w_j^2) \mid \mathrm{d}(w_i^1, w_j^2) \leq \epsilon\,\}$.

**Algorithm overview.** Algorithm 1 summarizes our technique. Block 1 randomly samples a small set of *seed* pairs, for which it explicly computes distances. The core of the algorithm (Block 3) explores pairs overlapping with a seed, looking for all appearance distances smaller than $\epsilon$. When

**Algorithm 1** Efficiently computing all distances smaller than $\epsilon$

---

Input: windows $\mathcal{W}^m = \{w_i^m\}$, threshold $\epsilon$, lookup table $\mathrm{o}_{min}$, number of initial samples $F$
Output: set $\mathcal{P}_\epsilon$ of all pairs $p$ with $\mathrm{d}(p) \leq \epsilon$

1. Compute seed pairs $\mathcal{P}_F$
   (a) sample $F$ random pairs $p_{ij} = (w_i^1, w_j^2)$ from $\mathcal{P} = \mathcal{W}^1 \times \mathcal{W}^2$, giving $\mathcal{P}_F$
   (b) **compute** $d_{ij} = \mathrm{d}(w_i^1, w_j^2), \ \ \forall p_{ij} \in \mathcal{P}_F$

2. Determine a sequence $S$ of all pairs from $\mathcal{P}$ (gives schedule of block 3 below)
   (a) sort the seed pairs in $\mathcal{P}_F$ in order of decreasing distance
   (b) set $S(1 : F) = \mathcal{P}_F$
   (c) fill $S((F + 1) : \mathrm{end})$ with random pairs from $\mathcal{P} \setminus \mathcal{P}_F$

3. For $p_c = S(1 : \mathrm{end})$ (explore the pairs in the $S$ order)
   (a) **compute** $d(p_c)$
   (b) if $d(p_c) \leq \epsilon$
       i. let $r = \mathrm{o}_{min}(\epsilon - d(p_c))$
       ii. let $\mathcal{N}$ = overlap_neighborhood($p_c$, $r$)
       iii. for all pairs $p \in \mathcal{N}$: **compute** $d(p)$
       iv. update $\mathcal{P}_\epsilon \leftarrow \mathcal{P}_\epsilon \cup \{p \in \mathcal{N} \,|\, d(p) \leq \epsilon\}$
   (c) else
       i. let $r = \mathrm{o}_{min}(d(p_c) - \epsilon)$
       ii. let $\mathcal{N}$ = overlap_neighborhood($p_c$, $r$)
       iii. **discard** all pairs in $\mathcal{N}$ from $S$: $\ \ S \leftarrow S \setminus \mathcal{N}$

**overlap_neighborhood**
Input: pair $p_{ij} = (w_i^1, w_j^2)$, overlap radius $r$
Output: overlap neighborhood $\mathcal{N}$ of $p_{ij}$

$\mathcal{N} = \{ (w_i^1, w_v^2) \,|\, \mathrm{o}(w_j^2, w_v^2) \geq r \} \cup \{(w_u^1, w_j^2) \,|\, \mathrm{o}(w_i^1, w_u^1) \geq r \}$

**compute**
Input: pair $p_{ij}$
Output: If $d(w_i^1, w_j^2)$ was never computed before, then compute it and store it in a table $D$. If $d(w_i^1, w_j^2)$ is already in $D$, then directly return it.

---

exploring a seed, the algorithm can decide to discard many pairs overlapping with it, as the bound predicts that their distance cannot be lower than $\epsilon$. This causes the computational saving (step 3.c). Before starting Block 3, Block 2 establishes the sequence in which to explore the seeds, i.e. in order of decreasing distance. The remaining pairs are appended in random order afterwards.

**Algorithm core.** Block 3 takes one of two actions based on the distance of the pair $p_c$ currently being explored. If $d(p_c) \leq \epsilon$, then all pairs in the overlap neighborhood $\mathcal{N}$ of $p_c$ have distance smaller than $\epsilon$. This overlap neighborhood has a radius $r = \mathrm{o}_{min}(\epsilon - d(p_c))$ predicted by the bound lookup table $\mathrm{o}_{min}$ (fig. 4a). Therefore, Block 3 computes the distance of all pairs in $\mathcal{N}$ (step 3.b). Instead, if $d(p_c) > \epsilon$, Block 3 determines the radius $r = \mathrm{o}_{min}(d(p_c) - \epsilon)$ of the overlap neighborhood containing pairs with distance greater than $\epsilon$, and then discards all pairs in it (step 3.c).

**Overlap neighborhood.** The overlap neighborhood of a pair $p_{ij} = (w_i^1, w_j^2)$ with radius $r$ contains all pairs $(w_i^1, w_v^2)$ such that $\mathrm{o}(w_j^2, w_v^2) \geq r$, and all pairs $(w_u^1, w_j^2)$ such that $\mathrm{o}(w_i^1, w_u^1) \geq r$ (fig. 4a).

## 4 Efficiently computing the single window pair with the smallest distance

We give an algorithm to efficiently find the single pair of windows with the smallest appearance distance between two images. Given as input the two sets of windows $\mathcal{W}^1, \mathcal{W}^2$, the algorithm should return the pair $p_* = (w_{i*}^1, w_{j*}^2)$ with the smallest distance: $\mathrm{d}(w_{i*}^1, w_{j*}^2) = \min_{ij} \mathrm{d}(w_i^1, w_j^2)$.

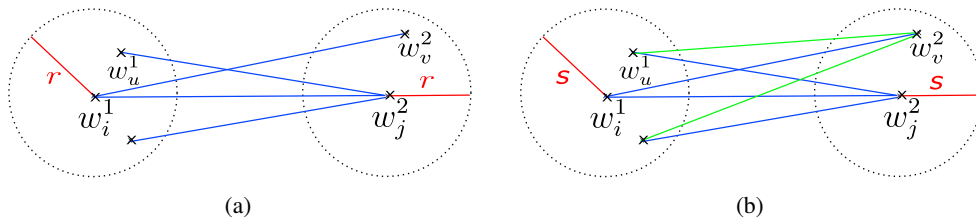

Fig. 4: **Overlap neighborhoods.** *(a) The overlap neighborhood of radius $r$ of a pair $(w_i^1, w_j^2)$ contains all blue pairs. (b) The joint overlap neighborhood of radius $s$ of a pair $(w_i^1, w_j^2)$ contains all blue and green pairs.*

**Algorithm overview.**    Algorithm 2 is analog to Algorithm 1. Block 1 computes distances for the seed pairs and it selectes the pair with the smallest distance as initial approximation to $p_*$. Block 3 explores pairs overlapping with a seed, looking for a distance smaller than $d(p_*)$. When exploring a seed, the algorithm can decide to discard many pairs overlapping with it, as the bound predicts they cannot be better than $p_*$. Block 2 organizes the seeds in order of increasing distance. In this way, the algorithm can rapidly refine $p_*$ towards smaller and smaller values. This is useful because in step 3.c, the amount of discarded pairs is greater as $d(p_*)$ gets smaller. Therefore, this seed ordering maximises the number of discarded pairs (i.e. minimizes the number of distances computed).

**Algorithm core.**    Block 3 takes one of two actions based on $d(p_c)$. If $d(p_c) \leq d(p_*) + \mathcal{B}_\alpha(s)$, then there might be a better pair than $d(p_*)$ within radius $s$ in the joint overlap neighborhood of $p_c$. Therefore, the algorithm computes the distance of all pairs in this neighborhood (step 3.b). The radius $s$ is an input parameter. Instead, if $d(p_c) > d(p_*) + \mathcal{B}_\alpha(s)$, the algorithm determines the radius $r = \mathrm{o}_{min}(d(p_c) - d(p_*))$ of the overlap neighborhood that contains only pairs with distance greater than $d(p_*)$, and then discards all pairs in it (step 3.c).

**Joint overlap neighborhood.**    The *joint* overlap neighborhood of a pair $p_{ij} = (w_i^1, w_j^2)$ with radius $s$ contains all pairs $(w_u^1, w_v^2)$ such that $\mathrm{o}(w_i^1, w_u^1) \geq s$ and $\mathrm{o}(w_j^2, w_v^2) \geq s$.

## 5  Experiments and conclusions

We present experiments on a test set composed of 1000 image pairs from the PASCAL VOC 07 dataset [12], randomly sampled under the constraint that two images in a pair contain at least one object of the same class (out of 6 classes: aeroplane, bicycle, bus, boat, horse, motorbike). This setting is relevant for various applications, such as object detection [13, 24], and ensures a balanced distribution of appearance distances in each image pair (some pairs of windows will have a low distance while others high distances). We experiment with three appearance descriptors: GIST [19] (960D), color histograms (CHIST, 4000D), and bag-of-words [11, 25] on the dense SURF descriptor [3] (BOW, 2000D). As appearance distances we use the Euclidean for GIST, and $\chi^2$ for CHIST and SURF BOW. The bound tables $\mathcal{B}_\alpha$ for each descriptor were estimated beforehand from a separate set of 1300 images of other classes (sec. 2.1).

**Task 1: all pairs of windows with distance smaller than $\epsilon$.**    The task is to find all pairs of windows with distance smaller than a user-defined threshold $\epsilon$ between two images $I^1, I^2$ (sec. 3). This task occurs in weakly supervised learning of object classes [7, 11, 16], where algorithms search for recurring patterns over training images containing thousands of overlapping windows, and in human pose estimation [22], which compares many overlapping candidate body part locations.

We random sample 3000 windows in each image ($|\mathcal{W}^1| = |\mathcal{W}^2| = 3000$) and set $\epsilon$ so that 10% of all distances are below it. This makes the task meaningful for any image pair, regardless of the range of distances it contains. For each image pair we quantify performance with two measures: (i) *cost*: the number of computed distances divided by the total number of window pairs (9 millions); (i) *accuracy*: $\frac{\sum_{p \in \mathcal{P}_\epsilon}(\epsilon - \mathrm{d}(p))}{\sum_{\{p \in \mathcal{W}^1 \times \mathcal{W}^2 | \mathrm{d}(p) \leq \epsilon\}}(\epsilon - \mathrm{d}(p))}$, where $\mathcal{P}_\epsilon$ is the set of window pairs returned by the algorithm, and the denominator sums over all distances truly below $\epsilon$. The lowest possible cost while still achieving 100% accuracy is 10%.

We compare to LSH [2, 6, 10] using [21] as a hash function. It maps descriptors to binary strings, such that the Hamming distance between two strings is related to the value of a Gaussian kernel between the original descriptors [21]. As recommended in [6, 10], we generate $T$ separate (random) encodings and build $T$ hash tables, each with $2^C$ bins, where $C$ is the number of bits in the encoding.

**Algorithm 2** Efficiently computing the smallest distance

Input: windows $\mathcal{W}^m = \{w_i^m\}$, lookup table $\mathrm{o}_{min}$, search radius $s$, number of initial samples $F$
Output: pair $p_*$ with the smallest distance

1. Compute seed pairs $\mathcal{P}_F$ (as Block 1 of Algorithm 1) and
   estimate current best pair: $p_* = \arg\min_{p_{ij} \in \mathcal{P}_F} d_{ij}$

2. Determine a sequence $S$ of all pairs (as Block 2 of Algorithm 1)

3. For $p_c = S(1 : \text{end})$   (explore the pairs in the $S$ order)

   (a) **compute** $d(p_c)$
   (b) if $d(p_c) \leq d(p_*) + \mathcal{B}_\alpha(s)$
       i. let $\mathcal{N}$ = joint_overlap_neighborhood($p_c, s$)
       ii. for all pairs $p \in \mathcal{N}$: **compute** $d(p)$
       iii. update $p_* \leftarrow \arg\min \{\{\mathrm{d}(p_*)\} \cup \{\mathrm{d}(p) \mid p \in \mathcal{N}\}\}$
   (c) else
       i. let $r = \mathrm{o}_{min}(d(p_c) - d(p_*))$
       ii. let $\mathcal{N}$ = overlap_neighborhood($p_c, r$)
       iii. **discard** all pairs in $\mathcal{N}$ from $S$:   $S \leftarrow S \setminus \mathcal{N}$

**joint_overlap_neighborhood**
Input pair $p_{ij} = (w_i^1, w_j^2)$, overlap radius $s$
Output: joint overlap neighborhood $\mathcal{N}$ of $p_{ij}$

$\mathcal{N} = \{ (w_u^1, w_v^2) \mid \mathrm{o}(w_i^1, w_u^1) \geq s, \ \mathrm{o}(w_j^2, w_v^2) \geq s \}$

To perform Task 1, we loop over each table $t$ and do: (H1) hash all $w_j^2 \in \mathcal{W}^2$ into table $t$; (H2) for each $w_i^1 \in \mathcal{W}^1$ do: (H2.1) hash $w_i^1$ into its bin $b_{t,i}^1$; (H2.2) compute all distances $\mathrm{d}$ in the original space between $w_i^1$ and all windows $w_j^2 \in b_{t,i}^1$ (unless already computed when inspecting a previous table); (H3) return all computed $\mathrm{d}(w_i^1, w_j^2) \leq \epsilon$.

We also compare to approximate nearest-neighbors based on kd-trees, using the ANN library [18]. To perform Task 1, we do: (A1) for each $w_i^1 \in \mathcal{W}^1$ do: (A1.1) compute the $\epsilon$-NN between $w_i^1$ and all windows $w_j^2 \in \mathcal{W}^2$ and return them all. The notion of cost above is not defined for ANN methods based on trees. Instead, we measure wall clock runtime. Instead, we report as cost the ratio of the runtime of approximate NN over the runtime of exact NN (also computed using the ANN library [18]). This gives a meaningful indication of speedup, which can be compared to the cost we report for our method and LSH. As the ANN library supports only the Euclidean distance, we report results only for GIST.

The results table reports cost and accuracy averaged over the test set. Our method from sec. 3 performs very well for all three descriptors. On average it achieves 98% accuracy at 16% cost. This is a considerable speedup over exhaustive search, as it means only 7% of the 90% distances greater than $\epsilon$ have been computed. The behavior of LSH depends on $T$ and $C$. The higher the $T$, the higher the accuracy, but also the cost (because there are more collisions; the same holds for lower $C$). To compare fairly, we evaluate LSH over $T \in \{1, 20\}$ and $C \in \{2, 30\}$ and report results for the $T, C$ that deliver the closest accuracy to our method. As the table shows, on average over the three descriptors, for same accuracy LSH has cost 92%, substantially worse than our method. The behavior of ANN depends on the degree of approximation which we set so as to get accuracy closest to our method. At 92% accuracy, ANN has 72% of the runtime of exact NN. This shows that, if high accuracy is desired, ANN offers only a modest speedup (compared to our 18% cost for GIST).

**Task 2: all windows closer than $\epsilon$ to a query.**   This is a special case of Task 1, where $\mathcal{W}^1$ contains just one window. Hence, this becomes a $\epsilon$-nearest-neighbours task where $\mathcal{W}^1$ acts as a query and $\mathcal{W}^2$ as the retrieval database. This task occurs in many applications, e.g. object detectors based on kernel SVMs compare a support vector (query) to a large set of overlapping windows in the test image [13, 24]. As this is expensive, many detectors resort to linear kernels [9]. Our algorithms

| Task 1 | | | | | | | | |
|---|---|---|---|---|---|---|---|---|
| GIST + Euclidean distance | | | CHIST + $\chi^2$ distance | | | SURF BOW + $\chi^2$ distance | | |
| method | cost | accuracy | method | cost | accuracy | method | cost | accuracy |
| our | 18.0% | 97.3% | our | 15.7% | 97.7% | our | 15.2% | 98.5% |
| LSH | 86.2% | 95.4% | LSH | 93.7% | 97.2% | LSH | 96.8% | 98.5% |
| ANN | 71.8% | 91.9% | ANN | - | - | ANN | - | - |
| Task 2 | | | | | | | | |
| method | cost | accuracy | method | cost | accuracy | method | cost | accuracy |
| our | 30.2% | 87.1% | our | 30.3% | 96.2% | our | 28.6% | 94.0% |
| LSH | 73.4% | 83.5% | LSH | 96.9% | 95.1% | LSH | 88.7% | 92.1% |
| ANN | 72.6% | 87.7% | ANN | - | - | ANN | - | - |

| Task 3 | | | | | | | | | | | |
|---|---|---|---|---|---|---|---|---|---|---|---|
| method | cost | ratio | rank | method | cost | ratio | rank | method | cost | ratio | rank |
| our | 2.3% | 1.02 | 1.39 | our | 0.4% | 1.01 | 1.12 | our | 0.7% | 1.01 | 1.19 |
| LSH | 16.4% | 1.03 | 2.72 | LSH | 37.5% | 1.02 | 33.5 | LSH | 46.5% | 1.01 | 9.62 |
| ANN | 58.6% | 1.01 | 1.48 | ANN | - | - | - | ANN | - | - | - |

offer the option to use more complex kernels while retaining a practical speed. Other applications include tracking in video [4, 5] and image retrieval [20] (see beginning of sec. 1).

As the table shows, our method is somewhat less efficient than on Task 1. This makes sense, as it can only exploit overlap structure in one of the two input sets. Yet, for a similar accuracy it offers greater speedup than LSH and ANN.

**Task 3: single pair of windows with smallest distance.** The task is to find the single pair of windows with the smallest distance between $I^1$ and $I^2$, out of 3000 windows in each image (sec. 4), and has similar applications as Task 1.

We quantify performance with three measures: (i) *cost*: as in all other tasks. (ii) *distance ratio*: the ratio between the smallest distance returned by the algorithm and the true smallest distance. The best possible value is 1, and higher values are worse; (iii) *rank*: the rank of the returned distance among all 9 million.

To perform Task 3 with LSH, we simply modify step (H3) of the procedure given for Task 1 to: return the smallest distance among all those computed. To perform Task 3 with ANN we replace step (A1.1) with: compute the NN of $w_i^1$ in $\mathcal{W}^2$. At the end of loop (A1) return the smallest distance among all those computed.

As the table shows, on average over the three descriptors, our method from sec. 4 achieves a distance ratio of 1.01 at 1.1% cost, which is almost a $100\times$ faster than exhaustive search. The average rank of the returned distance is 1.25 out of 9 millions, which is almost a perfect result. When compared at a similar distance ratio, our method is considerably more efficient than LSH and ANN. LSH computes 33.3% of all distances, while ANN brings only a speedup of factor 2 over exact NN.

**Runtime considerations.** While we have measured only the number of computed appearance distances, our algorithms also compute spatial overlaps. Crucially, spatial overlaps are computed in the 4D geometric space, compared to $1000+$ dimensions for the appearance space. Therefore, computing spatial overlaps has negligible impact on the total runtime of the algorithms. In practice, when using 5000 windows per image with 4000D dense SURF BOW descriptors, the total runtime of our algorithms is 71s for Task 1 or 16s for Task 3, compared to 335s for exhaustive search. Importantly, the cost of computing the descriptors is small compared to the cost of evaluating distances, as it is roughly linear in the number of windows and can be implemented very rapidly. In practice, computing dense SURF BOW for 5000 windows in two images takes 5 seconds.

**Conclusions.** We have proposed efficient algorithms for computing distances of appearance descriptors between two sets of image windows, by taking advantage of the overlap structure in the sets. Our experiments demonstrate that these algorithms greatly reduce the number of appearance distances computed when solving several tasks relevant to computer vision and outperform LSH and ANN for these tasks. Our algorithms could be useful in various applications. For example, improving the spatial accuracy of weakly supervised learners [7, 11] by using thousands of windows per image, using more complex kernels and detecting more classes in kernel SVM object detectors [13, 24], and enabling image retrieval systems to search at the window level with any descriptor, rather than returning entire images or be constrained to bag-of-words descriptors [20]. To encourage these applications, we release our source code at http://www.vision.ee.ethz.ch/~calvin.

# References

[1] B. Alexe, V. Petrescu, and V. Ferrari. Exploiting spatial overlap to efficiently compute appearance distances between image windows - supplementary material. In *NIPS*, 2011. Also available at http://www.vision.ee.ethz.ch/ calvin/publications.html.

[2] A. Andoni and P. Indyk. Near-optimal hashing algorithms for approximate nearest neighbor in high dimensions. In *Communications of the ACM*, 2008.

[3] H. Bay, A. Ess, T. Tuytelaars, and L. van Gool. SURF: Speeded up robust features. *CVIU*, 110(3):346–359, 2008.

[4] C. Bibby and I. Reid. Robust real-time visual tracking using pixel-wise posteriors. In *ECCV*, 2008.

[5] S. Birchfield. Elliptical head tracking using intensity gradients and color histograms. In *CVPR*, 1998.

[6] O. Chum, J. Philbin, M. Isard, and A. Zisserman. Scalable near identical image and shot detection. In *CIVR*, 2007.

[7] O. Chum and A. Zisserman. An exemplar model for learning object classes. In *CVPR*, 2007.

[8] N. Dalal and B. Triggs. Histogram of Oriented Gradients for Human Detection. In *CVPR*, volume 2, pages 886–893, 2005.

[9] N. Dalal and B. Triggs. Histogram of oriented gradients for human detection. In *CVPR*, 2005.

[10] M. Datar, N. Immorlica, P. Indyk, and V. Mirrokni. Locality-sensitive hashing scheme based on p-stable distributions. In *SCG*, 2004.

[11] T. Deselaers, B. Alexe, and V. Ferrari. Localizing objects while learning their appearance. In *ECCV*, 2010.

[12] M. Everingham, L. Van Gool, C. Williams, J. Winn, and A. Zisserman. The PASCAL Visual Object Classes Challenge 2007 Results, 2007.

[13] H. Harzallah, F. Jurie, and C. Schmid. Combining efficient object localization and image classification. In *ICCV*, 2009.

[14] H. Jegou, M. Douze, and C. Schmid. Hamming embedding and weak geometric consistency for large-scale image search. In *ECCV*, 2008.

[15] Y. Ke and R. Sukthankar. Pca-sift: A more distinctive representation for local image descriptors. In *CVPR*, 2004.

[16] G. Kim and A. Torralba. Unsupervised detection of regions of interest using iterative link analysis. In *NIPS*, 2009.

[17] N. Kumar, L. Zhang, and S. Nayar. What is a good nearest neighbors algorithm for finding similar patches in images? In *ECCV*, 2008.

[18] D. M. Mount and S. Arya. Ann: A library for approximate nearest neighbor searching, August 2006.

[19] A. Oliva and A. Torralba. Modeling the shape of the scene: a holistic representation of the spatial envelope. *IJCV*, 42(3):145–175, 2001.

[20] J. Philbin, O. Chum, M. Isard, J. Sivic, and A. Zisserman. Object retrieval with large vocabularies and fast spatial matching. In *CVPR*, 2007.

[21] M. Raginski and S. Lazebnik. Locality sensitive binary codes from shift-invariant kernels. In *NIPS*, 2009.

[22] B. Sapp, A. Toshev, and B. Taskar. Cascaded models for articulated pose estimation. In *ECCV*, 2010.

[23] A. Torralba, R. Fergus, and Y. Weiss. Small codes and large image databases for recognition. In *CVPR*, 2008.

[24] A. Vedaldi, V. Gulshan, M. Varma, and A. Zisserman. Multiple kernels for object detection. In *ICCV*, 2009.

[25] J. Zhang, M. Marszalek, S. Lazebnik, and C. Schmid. Local features and kernels for classification of texture and object categories: a comprehensive study. *IJCV*, 2007.

